# Radial Basis Function Networks and Complexity Regularization in Function Learning

**Adam Krzyżak**
Department of Computer Science
Concordia University
Montreal, Canada
krzyzak@cs.concordia.ca

**Tamás Linder**
Dept. of Math. & Comp. Sci.
Technical University of Budapest
Budapest, Hungary
linder@inf.bme.hu

## Abstract

In this paper we apply the method of complexity regularization to derive estimation bounds for nonlinear function estimation using a single hidden layer radial basis function network. Our approach differs from the previous complexity regularization neural network function learning schemes in that we operate with random covering numbers and $l_1$ metric entropy, making it possible to consider much broader families of activation functions, namely functions of bounded variation. Some constraints previously imposed on the network parameters are also eliminated this way. The network is trained by means of complexity regularization involving empirical risk minimization. Bounds on the expected risk in terms of the sample size are obtained for a large class of loss functions. Rates of convergence to the optimal loss are also derived.

## 1  INTRODUCTION

Artificial neural networks have been found effective in learning input-output mappings from noisy examples. In this learning problem an unknown target function is to be inferred from a set of independent observations drawn according to some unknown probability distribution from the input-output space $\mathbb{R}^d \times \mathbb{R}$. Using this data set the learner tries to determine a function which fits the data in the sense of minimizing some given empirical loss function. The target function may or may not be in the class of functions which are realizable by the learner. In the case when the class of realizable functions consists of some class of artificial neural networks, the above problem has been extensively studied from different viewpoints.

In recent years a special class of artificial neural networks, the radial basis function (RBF) networks have received considerable attention. RBF networks have been shown to be the solution of the regularization problem in function estimation with certain standard smoothness functionals used as stabilizers (see [5], and the references therein). Universal

convergence of RBF nets in function estimation and classification has been proven by
Krzyżak *et al.* [6]. Convergence rates of RBF approximation schemes have been shown to
be comparable with those for sigmoidal nets by Girosi and Anzellotti [4]. In a recent paper
Niyogi and Girosi [9] studied the tradeoff between approximation and estimation errors and
provided an extensive review of the problem.

In this paper we consider one hidden layer RBF networks. We look at the problem of
choosing the size of the hidden layer as a function of the available training data by means
of complexity regularization. Complexity regularization approach has been applied to
model selection by Barron [1], [2] resulting in near optimal choice of sigmoidal network
parameters. Our approach here differs from Barron's in that we are using $l_1$ metric en-
tropy instead of the supremum norm. This allows us to consider a more general class
of activation function, namely the functions of bounded variation, rather than a restricted
class of activation functions satisfying a Lipschitz condition. For example, activations with
jump discontinuities are allowed. In our complexity regularization approach we are able
to choose the network parameters more freely, and no discretization of these parameters is
required. For RBF regression estimation with squared error loss, we considerably improve
the convergence rate result obtained by Niyogi and Girosi [9].

In Section 2 the problem is formulated and two results on the estimation error of complexity
regularized RBF nets are presented: one for general loss functions (Theorem 1) and a
sharpened version of the first one for the squared loss (Theorem 2). Approximation bounds
are combined with the obtained estimation results in Section 3 yielding convergence rates
for function learning with RBF nets.

## 2   PROBLEM FORMULATION

The task is to predict the value of a real random variable $Y$ upon the observation of an $\mathbb{R}^d$
valued random vector $X$. The accuracy of the predictor $f : \mathbb{R}^d \to \mathbb{R}$ is measured by the
expected risk
$$J(f) = \mathbf{E}L(f(X), Y),$$
where $L : \mathbb{R} \times \mathbb{R} \to \mathbb{R}^+$ is a nonnegative loss function. It will be assumed that there exists
a minimizing predictor $f^*$ such that
$$J(f^*) = \inf_f J(f).$$

A good predictor $f_n$ is to be determined based on the data $(X_1, Y_1), \ldots, (X_n, Y_n)$ which
are i.i.d. copies of $(X, Y)$. The goal is to make the expected risk $\mathbf{E}J(f_n)$ as small as
possible, while $f_n$ is chosen from among a given class $\mathcal{F}$ of candidate functions.

In this paper the set of candidate functions $\mathcal{F}$ will be the set of single-layer feedforward
neural networks with radial basis function activation units and we let $\mathcal{F} = \cup_{k=1}^\infty \mathcal{F}_k$, where
$\mathcal{F}_k$ is the family of networks with $k$ hidden nodes whose weight parameters satisfy certain
constraints. In particular, for radial basis functions characterized by a kernel $K : \mathbb{R}^+ \to \mathbb{R}$,
$\mathcal{F}_k$ is the family of networks
$$f(x) = \sum_{i=1}^k w_i K\left([x - c_i]^t A_i [x - c_i]\right) + w_0,$$
where $w_0, w_1 \ldots, w_k$ are real numbers called weights, $c_1, \ldots, c_k \in \mathbb{R}^d$, $A_i$ are nonnegative
definite $d \times d$ matrices, and $x^t$ denotes the transpose of the column vector $x$.

The complexity regularization principle for the learning problem was introduced by Vapnik
[10] and fully developed by Barron [1], [2] (see also Lugosi and Zeger [8]). It enables
the learning algorithm to choose the candidate class $\mathcal{F}_k$ automatically, from which is picks

the estimate function by minimizing the empirical error over the training data. Complexity regularization penalizes the large candidate classes, which are bound to have small approximation error, in favor of the smaller ones, thus balancing the estimation and approximation errors.

Let $\mathcal{F}$ be a subset of a space $\mathcal{X}$ of real functions over some set, and let $\rho$ be a pseudometric on $\mathcal{X}$. For $\epsilon > 0$ the *covering number* $N(\epsilon, \mathcal{F}, \rho)$ is defined to be the minimal number of closed $\epsilon$ balls whose union cover $\mathcal{F}$. In other words, $N(\epsilon, \mathcal{F}, \rho)$ is the least integer such that there exist $f_1, \ldots, f_N$ with $N = N(\epsilon, \mathcal{F}, \rho)$ satisfying

$$\sup_{f \in \mathcal{F}} \min_{1 \leq i \leq N} \rho(f, f_i) \leq \epsilon.$$

In our case, $\mathcal{F}$ is a family of real functions on $\mathbb{R}^m$, and for any two functions $f$ and $g$, $\rho$ is given by

$$\rho(f, g) = \frac{1}{n} \sum_{i=1}^{n} |f(z_i) - g(z_i)|,$$

where $z_1, \ldots, z_n$ are $n$ given points in $\mathbb{R}^m$. In this case we will use the notation $N(\epsilon, \mathcal{F}, \rho) = N(\epsilon, \mathcal{F}, z_1^n)$, emphasizing the dependence of the metric $\rho$ on $z_1^n = (z_1, \ldots, z_n)$. Let us define the families of functions $\mathcal{H}_k$, $k = 1, 2, \ldots$ by

$$\mathcal{H}_k = \{L(f(\cdot), \cdot) : f \in \mathcal{F}_k\}.$$

Thus each member of $\mathcal{H}_k$ maps $\mathbb{R}^{d+1}$ into $\mathbb{R}$. It will be assumed that for each $k$ we are given a finite, almost sure uniform upper bound on the random covering numbers $N(\epsilon, \mathcal{H}_k, Z_1^n)$, where $Z_1^n = ((X_1, Y_1), \ldots, (X_n, Y_n))$. We may assume without loss of generality that $N(\epsilon, \mathcal{H}_k)$ is monotone decreasing in $\epsilon$. Finally, assume that $L(f(X), Y)$ is uniformly almost surely bounded by a constant $B$, i.e.,

$$\mathbf{P}\{L(f(X), Y) \leq B\} = 1, \quad f \in \mathcal{F}_k, \ k = 1, 2, \ldots \tag{1}$$

The complexity penalty of the $k$th class for $n$ training samples is a nonnegative number $\Delta_{kn}$ satisfying

$$\Delta_{kn} \geq \sqrt{128 B^2 \frac{\log N(\Delta_{kn}/8, \mathcal{H}_k) + c_k}{n}}, \tag{2}$$

where the nonnegative constants $c_k$ satisfy $\sum_{k=1}^{\infty} e^{-c_k} \leq 1$. Note that since $N(\epsilon, \mathcal{H}_k)$ is nonincreasing in $\epsilon$, it is possible to choose such $\Delta_{kn}$ for all $k$ and $n$. The resulting complexity penalty optimizes the upper bound on the estimation error in the proof of Theorem 1 below. We can now define our estimate. Let

$$f_{kn} = \arg\min_{f \in \mathcal{F}_k} J_n(f) = \arg\min_{f \in \mathcal{F}_k} \frac{1}{n} \sum_{i=1}^{n} L(f(X_i), Y_i),$$

that is, $f_{kn}$ minimizes over $\mathcal{F}_k$ the empirical risk for $n$ training samples. The penalized empirical risk is defined for each $f \in \mathcal{F}_k$ as

$$\widehat{J}_n(f) = J_n(f) + \Delta_{kn}.$$

The estimate $f_n$ is then defined as the $f_{kn}$ minimizing the penalized empirical risk over all classes:

$$f_n = \arg\min_{f_{kn}: k \geq 1} \widehat{J}_n(f_{kn}). \tag{3}$$

We have the following theorem for the expected estimation error of the above complexity regularization scheme.

**Theorem 1** *For any $n$ and $k$ the complexity regularization estimate* (3) *satisfies*

$$\mathbf{E}J(f_n) - J(f^*) \leq \min_{k \geq 1} \left( R_{kn} + \inf_{f \in \mathcal{F}_k} J(f) - J(f^*) \right),$$

*where*

$$R_{kn} = \min_{u \geq 4\Delta_{kn}} \left( u + 9Be^{-nu^2/(512B^2)} \right).$$

Assuming without loss of generality that $\log N(\epsilon, \mathcal{H}_k) \geq 1$, it is easy to see that the choice

$$\Delta_{kn} = \sqrt{128B^2 \frac{\log N(B/\sqrt{n}, \mathcal{H}_k) + c_k}{n}} \tag{4}$$

satisfies (2).

## 2.1 SQUARED ERROR LOSS

For the special case when

$$L(x, y) = (x - y)^2$$

we can obtain a better upper bound. The estimate will be the same as before, but instead of (2), the complexity penalty $\Delta_{kn}$ now has to satisfy

$$\Delta_{kn} \geq C_1 \frac{\log N(\Delta_{kn}/C_2, \mathcal{F}_k) + c_k}{n}, \tag{5}$$

where $C_1 = 3499C^4$, $C_2 = 256C^3$, and $C = \max\{B, 1\}$. Here $N(\epsilon, \mathcal{F}_k)$ is a uniform upper bound on the random $l_1$ covering numbers $N(\epsilon, \mathcal{F}_k, X_1^n)$. Assume that the class $\mathcal{F} = \cup_k \mathcal{F}_k$ is convex, and let $\overline{\mathcal{F}}$ be the closure of $\mathcal{F}$ in $L^2(\mu)$, where $\mu$ denotes the distribution of $X$. Then there is a unique $\bar{f} \in \overline{\mathcal{F}}$ whose squared loss $J(\bar{f})$ achieves $\inf_{f \in \mathcal{F}} J(f)$. We have the following bound on the difference $\mathbf{E}J(f_n) - J(\bar{f})$.

**Theorem 2** *Assume that $\mathcal{F} = \cup_k \mathcal{F}_k$ is a convex set of functions, and consider the squared error loss. Suppose that $|f(x)| \leq B$ for all $x \in \mathbb{R}^d$ and $f \in \mathcal{F}$, and $\mathbf{P}(|Y| > B) = 0$. Then complexity regularization estimate with complexity penalty satisfying (5) gives*

$$\mathbf{E}J(f_n) - J(\bar{f}) \leq 2 \min_{k \geq 1} \left( \Delta_{kn} + \inf_{f \in \mathcal{F}_k} J(f) - J(\bar{f}) \right) + \frac{C_1}{2n}.$$

The proof of this result uses an idea of Barron [1] and a Bernstein-type uniform probability inequality recently obtained by Lee *et al.* [7].

## 3 RBF NETWORKS

We will consider radial basis function (RBF) networks with one hidden layer. Such a network is characterized by a kernel $K : \mathbb{R}^+ \to \mathbb{R}$. An RBF net of $k$ nodes is of the form

$$f(x) = \sum_{i=1}^{k} w_i K \left( [x - c_i]^t A_i [x - c_i] \right) + w_0, \tag{6}$$

where $w_0, w_1, \ldots, w_k$ are real numbers called weights, $c_1, \ldots, c_k \in \mathbb{R}^d$, and the $A_i$ are nonnegative definite $d \times d$ matrices. The $k$th candidate class $\mathcal{F}_k$ for the function estimation task is defined as the class of networks with $k$ nodes which satisfy the weight condition $\sum_{i=0}^{k} |w_i| \leq b$ for a fixed $b > 0$:

$$\mathcal{F}_k = \left\{ \sum_{i=1}^{k} w_i K \left( [x - c_i]^t A_i [x - c_i] \right) + w_0 : \sum_{i=0}^{k} |w_i| \leq b \right\}. \tag{7}$$

Let $L(x, y) = |x - y|^p$, and

$$J(f) = \mathbf{E}|f(X) - Y|^p, \tag{8}$$

where $1 \leq p < \infty$. Let $\mu$ denote the probability measure induced by $X$. Define $\overline{\mathcal{F}}$ to be the closure in $L^p(\mu)$ of the convex hull of the functions $\widehat{b}K([x - c]^t A[x - c])$ and the constant function $h(x) = 1$, $x \in \mathbb{R}^d$, where $|\widehat{b}| \leq b$, $c \in \mathbb{R}^d$, and $A$ varies over all nonnegative $d \times d$ matrices. That is, $\overline{\mathcal{F}}$ is the closure of $\mathcal{F} = \cup_k \mathcal{F}_k$, where $\mathcal{F}_k$ is given in (7). Let $g \in \overline{\mathcal{F}}$ be arbitrary. If we assume that $|K|$ is uniformly bounded, then by Corollary 1 of Darken *et al.* [3], we have for $1 \leq p \leq 2$ that

$$\inf_{f \in \mathcal{F}_k} \|f - g\|_{L^p(\mu)} = O(1/\sqrt{k}), \tag{9}$$

where $\|f - g\|_{L^p(\mu)}$ denotes the $L^p(\mu)$ norm $\left(\int |f - f^*|^p d\mu\right)^{1/p}$, and $\mathcal{F}_k$ is given in (7). The approximation error $\inf_{f \in \mathcal{F}_k} J(f) - J(f^*)$ can be dealt with using this result if the optimal $f^*$ happens to be in $\overline{\mathcal{F}}$. In this case, we obtain

$$\inf_{f \in \mathcal{F}_k} J(f) - J(f^*) = O(1/\sqrt{k})$$

for all $1 \leq p \leq 2$. Values of $p$ close to 1 are of great importance for robust neural network regression.

When the kernel $K$ has a bounded total variation, it can be shown that $N(\epsilon, \mathcal{H}_k) \leq (A_1/\epsilon)^{A_2 k}$, where the constants $A_1, A_2$ depend on $\sup_x |K(x)|$, the total variation $V$ of $K$, the dimension $d$, and on the the constant $b$ in the definition (7) of $\mathcal{F}_k$. Then, if $1 \leq p \leq 2$, the following consequence of Theorem 1 can be proved for $L^p$ regression estimation.

**Theorem 3** *Let the kernel $K$ be of bounded variation and assume that $|Y|$ is bounded. Then for $1 \leq p \leq 2$ the error (8) of the complexity regularized estimate satisfies*

$$\begin{aligned}
\mathbf{E}J(f_n) - J(f^*) &\leq \min_{k \geq 1} \left[ O\left(\sqrt{\frac{k \log n}{n}}\right) + O\left(\sqrt{\frac{1}{k}}\right) \right] \\
&= O\left(\left(\frac{\log n}{n}\right)^{1/4}\right).
\end{aligned}$$

For $p = 1$, i.e., for $L^1$ regression estimation, this rate is known to be optimal within the logarithmic factor.

For squared error loss $J(f) = \mathbf{E}(f(X) - Y)^2$ we have $f^*(x) = \mathbf{E}(Y|X = x)$. If $f^* \in \overline{\mathcal{F}}$, then by (9) we obtain

$$\inf_{f \in \mathcal{F}_k} J(f) - J(f^*) = O(1/k). \tag{10}$$

It is easy to check that the class $\cup_k \mathcal{F}_k$ is convex if the $\mathcal{F}_k$ are the collections of RBF nets defined in (7). The next result shows that we can get rid of the square root in Theorem 3.

**Theorem 4** *Assume that $K$ is of bounded variation. Suppose furthermore that $|Y|$ is a bounded random variable, and let $L(x, y) = (x - y)^2$. Then the complexity regularization RBF squared regression estimate satisfies*

$$\mathbf{E}J(f_n) - \inf_{f \in \mathcal{F}} J(f) \leq 2 \min_{k \geq 1} \left( \inf_{f \in \mathcal{F}_k} J(f) - \inf_{f \in \mathcal{F}} J(f) + O\left(\frac{k \log n}{n}\right) \right) + O\left(\frac{1}{n}\right).$$

If $f^* \in \overline{\mathcal{F}}$, this result and (10) give

$$
\begin{aligned}
\mathbf{E}J(f_n) - J(f^*) &\leq \min_{k \geq 1} \left[ O\left(\frac{k \log n}{n}\right) + O\left(\frac{1}{k}\right) \right] \\
&= O\left(\left(\frac{\log n}{n}\right)^{1/2}\right).
\end{aligned}
\tag{11}
$$

This result sharpens and extends Theorem 3.1 of Niyogi and Girosi [9] where the weaker $O\left(\sqrt{\frac{k \log n}{n}}\right) + O\left(\frac{1}{k}\right)$ convergence rate was obtained (in a PAC-like formulation) for the squared loss of Gaussian RBF network regression estimation. The rate in (11) varies linearly with dimension. Our result is valid for a very large class of RBF schemes, including the Gaussian RBF networks considered in [9]. Besides having improved on the convergence rate, our result has the advantage of allowing kernels which are not continuous, such as the window kernel.

The above convergence rate results hold in the case when there exists an $f^*$ minimizing the risk which is a member of the $L^p(\mu)$ closure of $\mathcal{F} = \cup \mathcal{F}_k$, where $\mathcal{F}_k$ is given in (7). In other words, $f^*$ should be such that for all $\epsilon > 0$ there exists a $k$ and a member $f$ of $\mathcal{F}_k$ with $\|f - f^*\|_{L^p(\mu)} < \epsilon$. The precise characterization of $\overline{\mathcal{F}}$ seems to be difficult. However, based on the work of Girosi and Anzellotti [4] we can describe a large class of functions that is *contained* in $\overline{\mathcal{F}}$.

Let $H(x, t)$ be a real and bounded function of two variables $x \in \mathbb{R}^d$ and $t \in \mathbb{R}^n$. Suppose that $\lambda$ is a signed measure on $\mathbb{R}^n$ with finite total variation $\|\lambda\|$. If $g(x)$ is defined as

$$
g(x) = \int_{\mathbb{R}^n} H(x, t) \lambda(dt),
$$

then $g \in L^p(\mu)$ for any probability measure $\mu$ on $\mathbb{R}^d$. One can reasonably expect that $g$ can be approximated well by functions $f(x)$ of the form

$$
f(x) = \sum_{i=1}^{k} w_i H(x, t_i),
$$

where $t_1, \ldots, t_k \in \mathbb{R}^n$ and $\sum_{i=1}^{k} |w_i| \leq \|\lambda\|$. The case $m = d$ and $H(x, t) = G(x - t)$ is investigated in [4], where a detailed description of function spaces arising from the different choices of the basis function $G$ is given. Niyogi and Girosi [9] extends this approach to approximation by convex combinations of translates and dilates of a Gaussian function. In general, we can prove the following

**Lemma 1** *Let*

$$
g(x) = \int_{\mathbb{R}^n} H(x, t) \lambda(dt),
\tag{12}
$$

*where $H(x, t)$ and $\lambda$ are as above. Define for each $k \geq 1$ the class of functions*

$$
\mathcal{G}_k = \left\{ f(x) = \sum_{i=1}^{k} w_i H(x, t_i) : \sum_{i=0}^{k} |w_i| \leq \|\lambda\| \right\}.
$$

*Then for any probability measure $\mu$ on $\mathbb{R}^d$ and for any $1 \leq p < \infty$, the function $g$ can be approximated in $L^p(\mu)$ arbitrarily closely by members of $\mathcal{G} = \cup \mathcal{G}_k$, i.e.,*

$$
\inf_{f \in \mathcal{G}_k} \|f - g\|_{L^p(\mu)} \to 0 \quad as \quad k \to \infty.
$$

To prove this lemma one need only slightly adapt the proof of Theorem 8.2 in [4], or in a more elementary way following the lines of the probabilistic proof of Theorem 1 of [6]. To apply the lemma for RBF networks considered in this paper, let $n = d^2 + d$, $t = (A, c)$, and $H(x, t) = K\left([x - c]^t A[x - c]\right)$. Then we obtain that $\overline{\mathcal{F}}$ contains all the functions $g$ with the integral representation

$$g(x) = \int_{\mathbf{R}^{d^2+d}} K\left([x - c]^t A[x - c]\right) \lambda(dc\, dA),$$

for which $\|\lambda\| \leq b$, where $b$ is the constraint on the weights as in (7).

## Acknowledgements

This work was supported in part by NSERC grant OGP000270, Canadian National Networks of Centers of Excellence grant 293 and OTKA grant F014174.

## References

[1] A. R. Barron. Complexity regularization with application to artificial neural networks. In G. Roussas, editor, *Nonparametric Functional Estimation and Related Topics*, pages 561–576. NATO ASI Series, Kluwer Academic Publishers, Dordrecht, 1991.

[2] A. R. Barron. Approximation and estimation bounds for artificial neural networks. *Machine Learning*, 14:115–133, 1994.

[3] C. Darken, M. Donahue, L. Gurvits, and E. Sontag. Rate of approximation results motivated by robust neural network learning. In *Proc. Sixth Annual Workshop on Computational Learning Theory*, pages 303–309. Morgan Kauffman, 1993.

[4] F. Girosi and G. Anzellotti. Rates of convergence for radial basis functions and neural networks. In R. J. Mammone, editor, *Artificial Neural Networks for Speech and Vision*, pages 97–113. Chapman & Hall, London, 1993.

[5] F. Girosi, M. Jones, and T. Poggio. Regularization theory and neural network architectures. *Neural Computation*, 7:219–267, 1995.

[6] A. Krzyżak, T. Linder, and G. Lugosi. Nonparametric estimation and classification using radial basis function nets and empirical risk minimization. *IEEE Transactions on Neural Networks*, 7(2):475–487, March 1996.

[7] W. S. Lee, P. L. Bartlett, and R. C. Williamson. Efficient agnostic learning of neural networks with bounded fan-in. to be published in *IEEE Transactions on Information Theory*, 1995.

[8] G. Lugosi and K. Zeger. Concept learning using complexity regularization. *IEEE Transactions on Information Theory*, 42:48–54, 1996.

[9] P. Niyogi and F. Girosi. On the relationship between generalization error, hypothesis complexity, and sample complexity for radial basis functions. *Neural Computation*, 8:819–842, 1996.

[10] V. N. Vapnik. *Estimation of Dependencies Based on Empirical Data*. Springer-Verlag, New York, 1982.

